# A Neural Network Model of 3-D Lightness Perception

Luiz Pessoa
Federal Univ. of Rio de Janeiro
Rio de Janeiro, RJ, Brazil
pessoa@cos.ufrj.br

William D. Ross
Boston University
Boston, MA 02215
bill@cns.bu.edu

## Abstract

A neural network model of 3-D lightness perception is presented which builds upon the FACADE Theory Boundary Contour System/Feature Contour System of Grossberg and colleagues. Early ratio encoding by retinal ganglion neurons as well as psychophysical results on constancy across different backgrounds (background constancy) are used to provide functional constraints to the theory and suggest a *contrast negation hypothesis* which states that ratio measures between coplanar regions are given more weight in the determination of lightness of the respective regions. Simulations of the model address data on lightness perception, including the coplanar ratio hypothesis, the Benary cross, and White's illusion.

## 1  INTRODUCTION

Our everyday visual experience includes surface color constancy. That is, despite 1) variations in scene lighting and 2) movement or displacement across visual contexts, the color of an object appears to a large extent to be the same. Color constancy refers, then, to the fact that surface color remains largely constant despite changes in the intensity and composition of the light reflected to the eyes from both the object itself and from surrounding objects. This paper discusses a neural network model of 3D lightness perception — i.e., only the achromatic or black to white dimension of surface color perception is addressed. More specifically, the problem of *background constancy* (see 2 above) is addressed and mechanisms to accomplish it in a system exhibiting *illumination constancy* (see 1 above) are proposed.

A landmark result in the study of lightness was an experiment reported by Wallach (1948) who showed that for a disk-annulus pattern, lightness is given by the ratio of disk and annulus luminances (i.e., independent of overall illumination); the

so-called ratio principle. In another study, Whittle and Challands (1969) had subjects perform brightness matches in a haploscopic display paradigm. A striking result was that subjects always matched decrements to decrements, or increments to increments, but never increments to decrements. Whittle and Challands' (1969) results provide psychophysical support to the notion that the early visual system codes luminance ratios and not absolute luminance. These psychophysical results are in line with results from neurophysiology indicating that cells at early stages of the visual system encode local luminance contrast (Shapley and Enroth-Cugell, 1984). Note that lateral inhibition mechanisms are sensitive to local ratios and can be used as part of the explanation of illumination constancy.

Despite the explanatory power of the ratio principle, and the fact that the early stages of the visual system likely code contrast, several experiments have shown that, in general, ratios are insufficient to account for surface color perception. Studies of background constancy (Whittle and Challands, 1969; Land and McCann, 1971; Arend and Spehar, 1993), of the role of 3-D spatial layout and illumination arrangement on lightness perception (e.g., Gilchrist, 1977) as well as many other effects, argue against the sufficiency of local contrast measures (e.g., Benary cross, White's, 1979 illusion). The neural network model presented here addresses these data using several fields of neurally plausible mechanisms of lateral inhibition and excitation.

## 2  FROM LUMINANCE RATIOS TO LIGHTNESS

The *coplanar ratio hypothesis* (Gilchrist, 1977) states that the lightness of a given region is determined predominantly in relation to other coplanar surfaces, and not by equally weighted relations to all retinally adjacent regions. We propose that in the determination of lightness, contrast measures between non-coplanar adjacent surfaces are partially negated in order to preserve background constancy.

Consider the Benary Cross pattern (input stimulus in Fig. 2). If the gray patch on the cross is considered to be at the same depth as the cross, while the other gray patch is taken to be at the same depth as the background (which is below the cross), the gray patch on the cross should look lighter (since its lightness is determined in *relation* to the black cross), and the other patch darker (since its lightness is determined in *relation* to the white background). White's (1979) illusion can be discussed in similar terms (see the input stimulus in Fig. 3).

The mechanisms presented below implement a process of *partial contrast negation* in which the initial retinal contrast code is modulated by depth information such that the retinal contrast consistent with the depth interpretation is maintained while the retinal contrast not supported by depth is negated or attenuated.

## 3  A FILLING-IN MODEL OF 3-D LIGHTNESS

Contrast/Filling-in models propose that initial measures of boundary contrast followed by spreading of neural activity within filling-in compartments produce a response profile isomorphic with the percept (Gerrits & Vendrik, 1970; Cohen & Grossberg, 1984; Grossberg & Todorović, 1988; Pessoa, Mingolla, & Neumann, 1995). In this paper we develop a neural network model of lightness perception in the tradition of contrast/filling-in theories. The neural network developed here is an extension of the Boundary Contour System/Feature Contour System (BCS/FCS) proposed by Cohen and Grossberg (1984) and Grossberg and Mingolla (1985) to explain 3-D lightness data.

A fundamental idea of the BCS/FCS theory is that lateral inhibition achieves illumination constancy but requires the recovery of lightness by the filling-in, or diffusion, of *featural* quality ("lightness" in our case). The final diffused activities correspond to lightness, which is the outcome of interactions between boundaries and featural quality, whereby boundaries control the process of filling-in by forming gates of variable resistance to diffusion.

*How can the visual system construct 3-D lightness percepts from contrast measures obtained by retinotopic lateral inhibition?* A mechanism that is easily instantiated in a neural model and provides a straightforward modification to the contrast/filling-in proposal of Grossberg and Todorović (1988) is the use of depth-gated filling-in. This can be accomplished through a pathway that modulates boundary strength for boundaries between surfaces or objects across depth. The use of permeable or "leaky" boundaries was also used by Grossberg and Todorović (1988) for 2-D stimuli. In the current usage, permeability is actively increased at depth boundaries to partially negate the contrast effect — since filling-in proceeds more freely — and thus preserve lightness constancy across backgrounds. Figure 1 describes the four computational stages of the system.

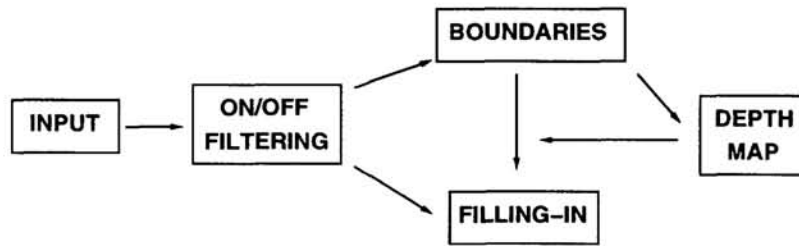

Figure 1: Model components.

**Stage 1: Contrast Measurement.** At this stage both ON and OFF neural fields with lateral inhibitory connectivity measure the strength of contrast at image regions — in uniform regions a contrast measurement of zero results. Formally, the ON field is given by

$$\frac{dy_{ij}^+}{dt} = -\alpha y_{ij}^+ + (\beta - y_{ij}^+)C_{ij}^+ - (y_{ij}^+ + \gamma)E_{ij}^+ \tag{1}$$

where $\alpha$, $\beta$ and $\gamma$ are constants; $C_{ij}^+$ is the total excitatory input to $y_{ij}^+$ and $E_{ij}^+$ is the total inhibitory input to $y_{ij}^+$. These terms denote discrete convolutions of the input $I_{ij}$ with Gaussian weighting functions, or kernels. An analogous equation specifies $y_{ij}^-$ for the OFF field. Figure 2 shows the ON-contrast minus the OFF-contrast.

**Stage 2: 2-D Boundary Detection.** At Stage 2, oriented odd-symmetric boundary detection cells are excited by the oriented sampling of the ON and OFF Stage 1 cells. Responses are maximal when ON activation is strong on one side of a cell's receptive field and OFF activation is strong on the opposite side. In other words, the cells are tuned to ON/OFF contrast co-occurrence, or juxtaposition (see Pessoa *et al.*, 1995). The output at this stage is the sum of the activations of such cells at each location for all orientations. The output responses are sharpened and localized through lateral inhibition across space; an equation similar to Equation 1 is used. The final output of Stage 2 is given by the signals $z_{ij}$ (see Fig. 2, Boundaries).

**Stage 3: Depth Map.** In the current implementation a simple scheme was employed for the determination of the depth configuration. Initially, four types of

T-junction cells detect such configurations in the image. For example,

$$T_{ij} = z_{i-d,j} \times z_{i+d,j} \times z_{i,j+d}, \tag{2}$$

where $d$ is a constant, detects T-junctions, where left, right, and top positions of the boundary stage are active; similar cells detect T-junctions of different orientations. The activities of the T-junction cells are then used in conjunction with boundary signals to define complete boundaries. Filling-in within these depth boundaries results in a depth map (see Fig. 2, Depth Map).

**Stage 4: Depth-modulated Filling-in**. In Stage 4, the ON and OFF contrast measures are allowed to diffuse across space within respective filling-in regions. Diffusion is blocked by boundary activations from Stage 2 (see Grossberg & Todorović, 1988, for details). The diffusion process is further modulated by depth information. The depth map provides this information; different activities code different depths. In a full blown implementation of the model, depth information would be obtained by the depth segmentation of the image supported by both binocular disparity and monocular depth cues.

Depth-modulated filling-in is such that boundaries across depths are reduced in strength. This allows a small percentage of the contrast on either side of the boundary to leak across it resulting in partial contrast negation, or reduction, at these boundaries. ON and OFF filling-in domains are used which receive the corresponding ON and OFF contrast activities from Stage 1 as inputs (see Fig. 2, Filled-in).

## 4  SIMULATIONS

The present model can account for several important phenomena, including 2 - D effects of lightness constancy and contrast (see Grossberg and Todorović, 1988). The simulations that follow address 3 -D lightness effects.

### 4.1  Benary Cross

Figure 2 shows the simulation for the Benary Cross. The plotted gray level values for filling-in reflect the activities of the ON filling-in domain minus the OFF domain. The model correctly predicts that the patch on the cross appears lighter than the patch on the background. This result is a direct consequence of contrast negation. The depth relationships are such that the patch on the cross is at the same depth as the cross and the patch on the background is at the same depth as the background (see Fig. 2, Depth Map). Therefore, the ratio of the background to the patch on the cross (across a depth boundary) and the ratio of the cross to the patch on the background (also across a depth boundary), are given a smaller weight in the lightness computation. Thus, the background will have a stronger effect on the appearance of the patch on the background, which will appear darker. At the same time, the cross will have a greater effect on the appearance of the patch on the cross, which will appear lighter.

### 4.2  White's Illusion

White's (1979) illusion (Fig. 3) is such that the gray patches on the black stripes appear lighter than the gray patches on the white stripes. This effect is considered a puzzling violation of *simultaneous contrast* since the contour length of the gray patches is larger for the stripes they do not lie on. Simultaneous contrast would predict that the gray patches on the black stripes appear lighter than the ones on white.

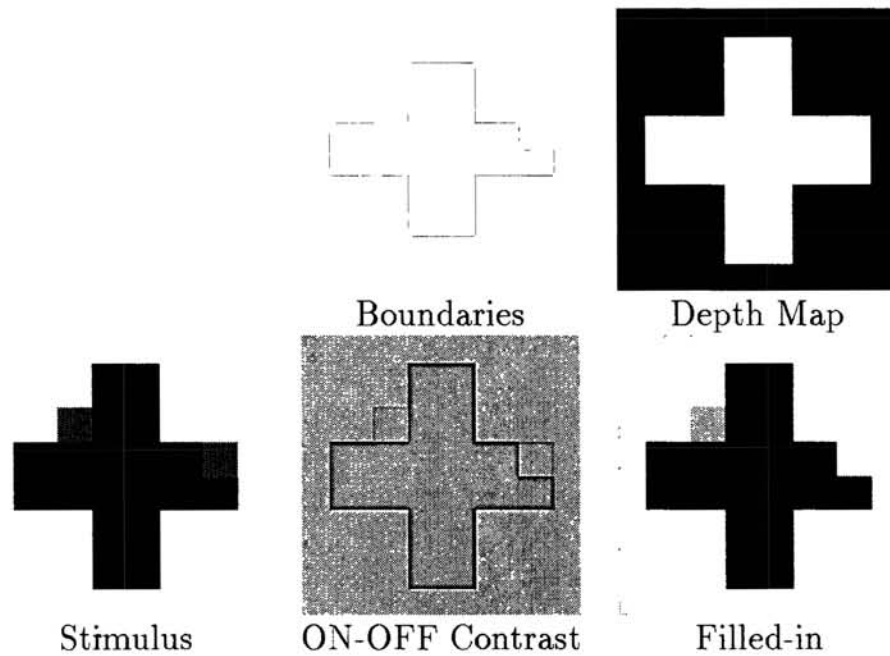

Figure 2: Benary Cross. The filled-in values of the gray patch on the cross are higher than the ones for the gray patch on the background. Gray levels code intensity; darker grays code lower values, lighter grays code higher values.

Figure 3 shows the result of the model for White's effect. The T-junction information in the stimulus determines that the gray patches are coplanar with the patches they lie on. Therefore, their appearance will be determined in relation to the contrast of their respective backgrounds. This is obtained, again, through contrast modulation, where the contrast of, say, the gray patch on a black stripe is preserved, while the contrast of the same patch with the white is partially negated (due to the depth arrangement).

### 4.3  Coplanar Hypothesis

Gilchrist (1977) showed that the perception of lightness is *not* determined by retinal adjacency, and that depth configuration and spatial layout help specify lightness. More specifically, it was proposed that the ratio of coplanar surfaces, not necessarily retinally adjacent, determines lightness, the so-called coplanar ratio hypothesis. Gilchrist was able to convincingly demonstrate this by comparing the perception of lightness in two equivalent displays (in terms of luminance values), aside from the *perceived* depth relationships in the displays.

Figure 4 shows computer simulations of the coplanar ratio effect. The same stimulus is given as input in two simulations with different depth specifications. In one (Depth Map 1), the depth map specifies that the rightmost patch is at a different depth than the two leftmost patches which are coplanar. In the other (Depth Map 2), the two rightmost patches are coplanar and at a different depth than the leftmost patch. In all, the depth organization alters the lightness of the central region, which should appear darker in the configuration of Depth Map 1 than the one for Depth Map 2. For Depth Map 1, since the middle patch is coplanar with a white patch, this patch is darkened by simultaneous contrast. For Depth Map 2, the middle patch will be lightened by contrast since it is coplanar with a black patch. It should be noted that the depth maps for the simulations shown in Fig. 4 were given as input.

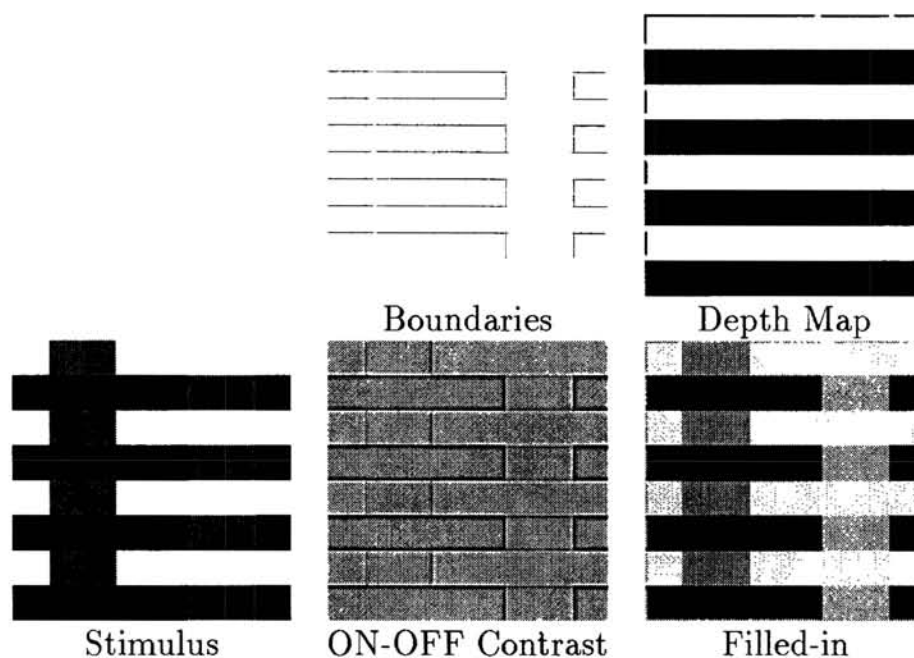

Figure 3: White's effect. The filled-in values of the gray patches on the black stripes are higher than the ones for the gray patches on white stripes.

The current implementation cannot recover depth trough binocular disparity and only employs monocular cues as in the previous simulations.

## 5   CONCLUSIONS

In this paper, data from experiments on lightness perception were used to extend the BCS/FCS theory of Grossberg and colleagues to account for several challenging phenomena. The model is an initial step towards providing an account that can take into consideration the complex factors involved in 3-D vision — see Grossberg (1994) for a comprehensive account of 3-D vision.

### Acknowledgements

The authors would like to than Alan Gilchrist and Fred Bonato for their suggestions concerning this work. L. P. was supported in part by Air Force Office of Scientific Research (AFOSR F49620-92-J-0334) and Office of Naval Research (ONR N00014-91-J-4100); W. R. was supported in part by HNC SC-94-001.

## Reference

Arend, L., & Spehar, B. (1993) Lightness, brightness, and brightness contrast: 2. Reflectance variation. *Perception & Psychophysics* 54:4576-468.

Cohen, M., & Grossberg, S. (1984) Neural dynamics of brightness perception: Features, boundaries, diffusion, and resonance. *Perception & Psychophysics* 36:428-456.

Gerrits, H. & Vendrik, A. (1970) Simultaneous contrast, filling-in process and information processing in man's visual system. *Experimental Brain Research* 11:411-430.

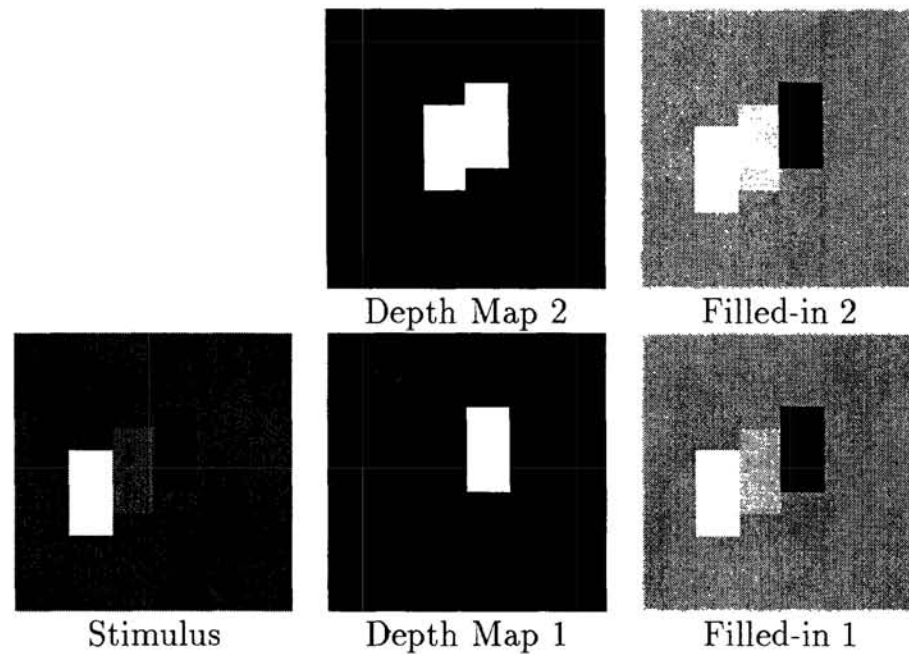

Figure 4: Gilchrist's coplanarity. The Filled-in values for the middle patch on top are higher than on bottom.

Gilchrist, A. (1977) Perceived lightness depends on perceived spatial arrangement. *Science* **195**:185-187.

Grossberg, S. (1994) 3-D vision and figure-ground separation by visual cortex. *Perception & Psychophysics* **55**:48-120.

Grossberg, S., & Mingolla, E. (1985) Neural dynamics of form perception: Boundary completion, illusory figures, and neon color spreading. *Psychological Review* **92**:173-211.

Grossberg, S., & Todorović. D. (1988). Neural dynamics of 1-D and 2-D brightness perception: A unified model of classical and recent phenomena. *Perception & Psychophysics* **43**:241-277.

Land, E., & McCann, J. (1971). Lightness and retinex theory. *Journal of the Optical Society of America* **61**:1-11.

Pessoa, L., Mingolla, E., & Neumann, H. (1995) A contrast- and luminance-driven multiscale network model of brightness perception. *Vision Research* **35**:2201-2223.

Shapley, R., & Enroth-Cugell, C. (1984) Visual adaptation and retinal gain controls. In N. Osborne and G. Chader (eds.), *Progress in Retinal Research*, pp. 263-346. Oxford: Pergamon Press.

Wallach, H. (1948) Brightness constancy and the nature of achromatic colors. *Journal of Experimental Psychology* **38**: 310-324.

White, M. (1979) A new effect of pattern on perceived lightness. *Perception* **8**:413-416.

Whittle, P., & Challands, P. (1969) The effect of background luminance on the brightness of flashes. *Vision Research* **9**:1095-1110.
